# An Impossibility Theorem for Clustering

**Jon Kleinberg**
Department of Computer Science
Cornell University
Ithaca NY 14853

## Abstract

Although the study of *clustering* is centered around an intuitively compelling goal, it has been very difficult to develop a unified framework for reasoning about it at a technical level, and profoundly diverse approaches to clustering abound in the research community. Here we suggest a formal perspective on the difficulty in finding such a unification, in the form of an *impossibility theorem*: for a set of three simple properties, we show that there is no clustering function satisfying all three. Relaxations of these properties expose some of the interesting (and unavoidable) trade-offs at work in well-studied clustering techniques such as single-linkage, sum-of-pairs, $k$-means, and $k$-median.

## 1 Introduction

Clustering is a notion that arises naturally in many fields; whenever one has a heterogeneous set of objects, it is natural to seek methods for grouping them together based on an underlying measure of similarity. A standard approach is to represent the collection of objects as a set of abstract points, and define distances among the points to represent similarities — the closer the points, the more similar they are. Thus, clustering is centered around an intuitively compelling but vaguely defined goal: given an underlying set of points, partition them into a collection of *clusters* so that points in the same cluster are close together, while points in different clusters are far apart.

The study of clustering is unified only at this very general level of description, however; at the level of concrete methods and algorithms, one quickly encounters a bewildering array of different clustering techniques, including agglomerative, spectral, information-theoretic, and centroid-based, as well as those arising from combinatorial optimization and from probabilistic generative models. These techniques are based on diverse underlying principles, and they often lead to qualitatively different results. A number of standard textbooks [1, 4, 6, 9] provide overviews of a range of the approaches that are generally employed.

Given the scope of the issue, there has been relatively little work aimed at reasoning about clustering independently of any particular algorithm, objective function, or generative data model. But it is not clear that this needs to be the case. To take a motivating example from a technically different but methodologically similar set-

ting, research in mathematical economics has frequently formalized broad intuitive notions (how to fairly divide resources, or how to achieve consensus from individual preferences) in what is often termed an *axiomatic* framework — one enumerates a collection of simple properties that a solution ought to satisfy, and then studies how these properties constrain the solutions one is able to obtain [10]. In some striking cases, as in Arrow's celebrated theorem on social choice functions [2], the result is *impossibility* — there is no solution that simultaneously satisfies a small collection of simple properties.

In this paper, we develop an axiomatic framework for clustering. First, as is standard, we define a *clustering function* to be any function $f$ that takes a set $S$ of $n$ points with pairwise distances between them, and returns a partition of $S$. (The points in $S$ are not assumed to belong to any ambient space; the pairwise distances are the only data one has about them.) We then consider the effect of requiring the clustering function to obey certain natural properties. Our first result is a basic impossibility theorem: for a set of three simple properties — essentially *scale-invariance*, a *richness* requirement that all partitions be achievable, and a *consistency* condition on the shrinking and stretching of individual distances — we show that there is no clustering function satisfying all three. None of these properties is redundant, in the sense that it is easy to construct clustering functions satisfying any two of the three. We also show, by way of contrast, that certain natural relaxations of this set of properties are satisfied by versions of well-known clustering functions, including those derived from single-linkage and sum-of-pairs. In particular, we fully characterize the set of possible outputs of a clustering function that satisfies the scale-invariance and consistency properties.

How should one interpret an impossibility result in this setting? The fact that it arises directly from three simple constraints suggests a technical underpinning for the difficulty in unifying the initial, informal concept of "clustering." It indicates a set of basic trade-offs that are inherent in the clustering problem, and offers a way to distinguish between clustering methods based not simply on operational grounds, but on the ways in which they resolve the choices implicit in these trade-offs. Exploring relaxations of the properties helps to sharpen this type of analysis further — providing a perspective, for example, on the distinction between clustering functions that fix the number of clusters *a priori* and those that do not; and between clustering functions that build in a fundamental length scale and those that do not.

**Other Axiomatic Approaches.** As discussed above, the vast majority of approaches to clustering are derived from the application of specific algorithms, the optima of specific objective functions, or the consequences of particular probabilistic generative models for the data. Here we briefly review work seeking to examine properties that do not overtly impose a particular objective function or model.

Jardine and Sibson [7] and Puzicha, Hofmann, and Buhmann [12] have considered axiomatic approaches to clustering, although they operate in formalisms quite different from ours, and they do not seek impossibility results. Jardine and Sibson are concerned with hierarchical clustering, where one constructs a tree of nested clusters. They show that a hierarchical version of single-linkage is the unique function consistent with a collection of properties; however, this is primarily a consequence of the fact that one of their properties is an implicit optimization criterion that is uniquely optimized by single-linkage. Puzicha et al. consider properties of cost functions on partitions; these implicitly define clustering functions through the process of choosing a minimum-cost partition. They investigate a particular class of clustering functions that arises if one requires the cost function to decompose into a certain additive form. Recently, Kalai, Papadimitriou, Vempala, and Vetta have

also investigated an axiomatic framework for clustering [8]; like the approach of Jardine and Sibson [7], and in contrast to our work here, they formulate a collection of properties that are sufficient to uniquely specify a particular clustering function.

Axiomatic approaches have also been applied in areas related to clustering — particularly in collaborative filtering, which harnesses similarities among users to make recommendations, and in discrete location theory, which focuses on the placement of "central" facilities among distributed collections of individuals. For collaborative filtering, Pennock et al. [11] show how results from social choice theory, including versions of Arrow's Impossibility Theorem [2], can be applied to characterize recommendation systems satisfying collections of simple properties. In discrete location theory, Hansen and Roberts [5] prove an impossibility result for choosing a central facility to serve a set of demands on a graph; essentially, given a certain collection of required properties, they show that any function that specifies the resulting facility must be highly sensitive to small changes in the input.

## 2  The Impossibility Theorem

A clustering function operates on a set $S$ of $n \geq 2$ points and the pairwise distances among them. Since we wish to deal with point sets that do not necessarily belong to an ambient space, we identify the points with the set $S = \{1, 2, \ldots, n\}$. We then define a *distance function* to be any function $d : S \times S \to \mathbf{R}$ such that for distinct $i, j \in S$, we have $d(i, j) \geq 0$, $d(i, j) = 0$ if and only if $i = j$, and $d(i, j) = d(j, i)$. One can optionally restrict attention to distance functions that are *metrics* by imposing the triangle inequality: $d(i, k) \leq d(i, j) + d(j, k)$ for all $i, j, k \in S$. We will not require the triangle inequality in the discussion here, but the results to follow — both negative and positive — still hold if one does require it.

A *clustering function* is a function $f$ that takes a distance function $d$ on $S$ and returns a partition $\Gamma$ of $S$. The sets in $\Gamma$ will be called its *clusters*. We note that, as written, a clustering function is defined only on point sets of a particular size $(n)$; however, all the specific clustering functions we consider here will be defined for all values of $n$ larger than some small base value.

Here is a first property one could require of a clustering function. If $d$ is a distance function, we write $\alpha \cdot d$ to denote the distance function in which the distance between $i$ and $j$ is $\alpha d(i, j)$.

> SCALE-INVARIANCE. *For any distance function $d$ and any $\alpha > 0$, we have $f(d) = f(\alpha \cdot d)$.*

This is simply the requirement that the clustering function not be sensitive to changes in the units of distance measurement — it should not have a built-in "length scale." A second property is that the output of the clustering function should be "rich" — every partition of $S$ is a possible output. To state this more compactly, let Range($f$) denote the set of all partitions $\Gamma$ such that $f(d) = \Gamma$ for some distance function $d$.

> RICHNESS. Range($f$) is equal to the set of all partitions of $S$.

In other words, suppose we are given the names of the points only (i.e. the indices in $S$) but not the distances between them. Richness requires that for any desired partition $\Gamma$, it should be possible to construct a distance function $d$ on $S$ for which $f(d) = \Gamma$.

Finally, we discuss a *Consistency* property that is more subtle that the first two. We think of a clustering function as being "consistent" if it exhibits the following behavior: when we shrink distances between points inside a cluster and expand distances between points in different clusters, we get the same result. To make this precise, we introduce the following definition. Let $\Gamma$ be a partition of $S$, and $d$ and $d'$ two distance functions on $S$. We say that $d'$ is a $\Gamma$-*transformation* of $d$ if (a) for all $i, j \in S$ belonging to the same cluster of $\Gamma$, we have $d'(i, j) \leq d(i, j)$; and (b) for all $i, j \in S$ belonging to different clusters of $\Gamma$, we have $d'(i, j) \geq d(i, j)$.

> CONSISTENCY. Let $d$ and $d'$ be two distance functions. If $f(d) = \Gamma$, and $d'$ is a $\Gamma$-transformation of $d$, then $f(d') = \Gamma$.

In other words, suppose that the clustering $\Gamma$ arises from the distance function $d$. If we now produce $d'$ by reducing distances within the clusters and enlarging distance between the clusters then the same clustering $\Gamma$ should arise from $d'$.

We can now state the impossibility theorem very simply.

**Theorem 2.1** *For each $n \geq 2$, there is no clustering function $f$ that satisfies Scale-Invariance, Richness, and Consistency.*

We will prove Theorem 2.1 in the next section, as a consequence of a more general statement. Before doing this, we reflect on the relation of these properties to one another by showing that there exist natural clustering functions satisfying any two of the three properties.

To do this, we describe the *single-linkage* procedure (see e.g. [6]), which in fact defines a family of clustering functions. Intuitively, single-linkage operates by initializing each point as its own cluster, and then repeatedly merging the pair of clusters whose distance to one another (as measured from their closest points of approach) is minimum. More concretely, single-linkage constructs a weighted complete graph $G_d$ whose node set is $S$ and for which the weight on edge $(i, j)$ is $d(i, j)$. It then orders the edges of $G_d$ by non-decreasing weight (breaking ties lexicographically), and adds edges one at a time until a specified *stopping condition* is reached. Let $H_d$ denote the subgraph consisting of all edges that are added before the stopping condition is reached; the connected components of $H_d$ are the clusters.

Thus, by choosing a stopping condition for the single-linkage procedure, one obtains a clustering function, which maps the input distance function to the set of connected components that results at the end of the procedure. We now show that for any two of the three properties in Theorem 2.1, one can choose a single-linkage stopping condition so that the resulting clustering function satisfies these two properties. Here are the three types of stopping conditions we will consider.

- *$k$-cluster stopping condition.* Stop adding edges when the subgraph first consists of $k$ connected components. (We will only consider this condition to be well-defined when the number of points is at least $k$.)

- *distance-$r$ stopping condition.* Only add edges of weight at most $r$.

- *scale-$\alpha$ stopping condition.* Let $\rho^*$ denote the maximum pairwise distance; i.e. $\rho^* = \max_{i,j} d(i, j)$. Only add edges of weight at most $\alpha \rho^*$.

It is clear that these various stopping conditions qualitatively trade off certain of the properties in Theorem 2.1. Thus, for example, the $k$-cluster stopping condition does not attempt to produce all possible partitions, while the distance-$r$ stopping condition builds in a fundamental length scale, and hence is not scale-invariant.

However, by the appropriate choice of one of these stopping conditions, one can achieve any two of the three properties in Theorem 2.1.

**Theorem 2.2** *(a) For any $k \geq 1$, and any $n \geq k$, single-linkage with the k-cluster stopping condition satisfies Scale-Invariance and Consistency.*

*(b) For any positive $\alpha < 1$, and any $n \geq 3$, single-linkage with the scale-$\alpha$ stopping condition satisfies Scale-Invariance and Richness.*

*(c) For any $r > 0$, and any $n \geq 2$, single-linkage with the distance-r stopping condition satisfies Richness and Consistency.*

## 3   Antichains of Partitions

We now state and prove a strengthening of the impossibility result. We say that a partition $\Gamma'$ is a *refinement* of a partition $\Gamma$ if for every set $C' \in \Gamma'$, there is a set $C \in \Gamma$ such that $C' \subseteq C$. We define a partial order on the set of all partitions by writing $\Gamma' \preceq \Gamma$ if $\Gamma'$ is a refinement of $\Gamma$. Following the terminology of partially ordered sets, we say that a collection of partitions is an *antichain* if it does not contain two distinct partitions such that one is a refinement of the other.

For a set of $n \geq 2$ points, the collection of all partitions does not form an antichain; thus, Theorem 2.1 follows from

**Theorem 3.1** *If a clustering function f satisfies Scale-Invariance and Consistency, then* $\mathrm{Range}(f)$ *is an antichain.*

*Proof.*   For a partition $\Gamma$, we say that a distance function $d$ $(a, b)$-*conforms* to $\Gamma$ if, for all pairs of points $i, j$ that belong to the same cluster of $\Gamma$, we have $d(i, j) \leq a$, while for all pairs of points $i, j$ that belong to different clusters, we have $d(i, j) \geq b$. With respect to a given clustering function $f$, we say that a pair of positive real numbers $(a, b)$ is $\Gamma$-*forcing* if, for all distance functions $d$ that $(a, b)$-conform to $\Gamma$, we have $f(d) = \Gamma$.

Let $f$ be a clustering function that satisfies Consistency. We claim that for any partition $\Gamma \in \mathrm{Range}(f)$, there exist positive real numbers $a < b$ such that the pair $(a, b)$ is $\Gamma$-forcing. To see this, we first note that since $\Gamma \in \mathrm{Range}(f)$, there exists a distance function $d$ such that $f(d) = \Gamma$. Now, let $a'$ be the minimum distance among pairs of points in the same cluster of $\Gamma$, and let $b'$ be the maximum distance among pairs of points that do not belong to the same cluster of $\Gamma$. Choose numbers $a < b$ so that $a \leq a'$ and $b \geq b'$. Clearly any distance function $d'$ that $(a, b)$-conforms to $\Gamma$ must be a $\Gamma$-transformation of $d$, and so by the Consistency property, $f(d') = \Gamma$. It follows that the pair $(a, b)$ is $\Gamma$-forcing.

Now suppose further that the clustering function $f$ satisfies Scale-Invariance, and that there exist distinct partitions $\Gamma_0, \Gamma_1 \in \mathrm{Range}(f)$ such that $\Gamma_0$ is a refinement of $\Gamma_1$. We show how this leads to a contradiction.

Let $(a_0, b_0)$ be a $\Gamma_0$-forcing pair, and let $(a_1, b_1)$ be a $\Gamma_1$-forcing pair, where $a_0 < b_0$ and $a_1 < b_1$; the existence of such pairs follows from our claim above. Let $a_2$ be any number less than or equal to $a_1$, and choose $\varepsilon$ so that $0 < \varepsilon < a_0 a_2 b_0^{-1}$. It is now straightforward to construct a distance function $d$ with the following properties: For pairs of points $i, j$ that belong to the same cluster of $\Gamma_0$, we have $d(i, j) \leq \varepsilon$; for pairs $i, j$ that belong to the same cluster of $\Gamma_1$ but not to the same cluster of $\Gamma_0$, we have $a_2 \leq d(i, j) \leq a_1$; and for pairs $i, j$ the do not belong to the same cluster of $\Gamma_1$, we have $d(i, j) \geq b_1$.

The distance function $d$ $(a_1, b_1)$-conforms to $\Gamma_1$, and so we have $f(d) = \Gamma_1$. Now set $\alpha = b_0 a_2^{-1}$, and define $d' = \alpha \cdot d$. By Scale-Invariance, we must have $f(d') = f(d) = \Gamma_1$. But for points $i, j$ in the same cluster of $\Gamma_0$ we have $d'(i, j) \leq \varepsilon b_0 a_2^{-1} < a_0$, while for points $i, j$ that do not belong to the same cluster of $\Gamma_0$ we have $d'(i, j) \geq a_2 b_0 a_2^{-1} \geq b_0$. Thus $d'$ $(a_0, b_0)$-conforms to $\Gamma_0$, and so we must have $f(d') = \Gamma_0$. As $\Gamma_0 \neq \Gamma_1$, this is a contradiction. ∎

The proof above uses our assumption that the clustering function $f$ is defined on the set of all distance functions on $n$ points. However, essentially the same proof yields a corresponding impossibility result for clustering functions $f$ that are defined only on metrics, or only on distance functions arising from $n$ points in a Euclidean space of some dimension. To adapt the proof, one need only be careful to choose the constant $a_2$ and distance function $d$ to satisfy the required properties.

We now prove a complementary positive result; together with Theorem 3.1, this completely characterizes the possible values of $\mathrm{Range}(f)$ for clustering functions $f$ that satisfy Scale-Invariance and Consistency.

**Theorem 3.2** *For every antichain of partitions $\mathcal{A}$, there is a clustering function $f$ satisfying Scale-Invariance and Consistency for which $\mathrm{Range}(f) = \mathcal{A}$.*

*Proof.* Given an arbitrary antichain $\mathcal{A}$, it is not clear how to produce a stopping condition for the single-linkage procedure that gives rise to a clustering function $f$ with $\mathrm{Range}(f) = \mathcal{A}$. (Note that the $k$-cluster stopping condition yields a clustering function whose range is the antichain consisting of all partitions into $k$ sets.) Thus, to prove this result, we use a variant of the *sum-of-pairs* clustering function (see e.g. [3]), adapted to general antichains. We focus on the case in which $|\mathcal{A}| > 1$, since the case of $|\mathcal{A}| = 1$ is trivial.

For a partition $\Gamma \in \mathcal{A}$, we write $(i, j) \sim \Gamma$ if both $i$ and $j$ belong to the same cluster in $\Gamma$. The $\mathcal{A}$-*sum-of-pairs* function $f$ seeks the partition $\Gamma \in \mathcal{A}$ that minimizes the sum of all distances between pairs of points in the same cluster; in other words, it seeks the $\Gamma \in \mathcal{A}$ minimizing the objective function $\Phi_d(\Gamma) = \sum_{(i,j) \sim \Gamma} d(i, j)$. (Ties are broken lexicographically.) It is crucial that the minimization is only over partitions in $\mathcal{A}$; clearly, if we wished to minimize this objective function over *all* partitions, we would choose the partition in which each point forms its own cluster.

It is clear that $f$ satisfies Scale-Invariance, since $\Phi_{\alpha \cdot d}(\Gamma) = \alpha \Phi_d(\Gamma)$ for any partition $\Gamma$. By definition we have $\mathrm{Range}(f) \subseteq \mathcal{A}$, and we argue that $\mathrm{Range}(f) \supseteq \mathcal{A}$ as follows. For any partition $\Gamma \in \mathcal{A}$, construct a distance function $d$ with the following properties: $d(i, j) < n^{-3}$ for every pair of points $i, j$ belonging to the same cluster of $\Gamma$, and $d(i, j) \geq 1$ for every pair of points $i, j$ belonging to different clusters of $\Gamma$. We have $\Phi_d(\Gamma) < 1$; and moreover $\Phi_d(\Gamma') < 1$ only for partitions $\Gamma'$ that are refinements of $\Gamma$. Since $\mathcal{A}$ is an antichain, it follows that $\Gamma$ must minimize $\Phi_d$ over all partitions in $\mathcal{A}$, and hence $f(d) = \Gamma$.

It remains only to verify Consistency. Suppose that for the distance function $d$, we have $f(d) = \Gamma$; and let $d'$ be a $\Gamma$-transformation of $d$. For any partition $\Gamma'$, let $\Delta(\Gamma') = \Phi_d(\Gamma') - \Phi_{d'}(\Gamma')$. It is enough to show that for any partition $\Gamma' \in \mathcal{A}$, we have $\Delta(\Gamma) \geq \Delta(\Gamma')$.

But this follows simply because $\Delta(\Gamma) = \sum_{(i,j) \sim \Gamma} d(i, j) - d'(i, j)$, while

$$\Delta(\Gamma') = \sum_{(i,j) \sim \Gamma'} d(i, j) - d'(i, j) \leq \sum_{(i,j) \sim \Gamma' \text{ and } (i,j) \sim \Gamma} d(i, j) - d'(i, j) \leq \Delta(\Gamma),$$

where both inequalities follow because $d'$ is a $\Gamma$-transformation of $d$: first, only

terms corresponding to pairs in the same cluster of $\Gamma$ are non-negative; and second, every term corresponding to a pair in the same cluster of $\Gamma$ is non-negative. ■

## 4    Centroid-Based Clustering and Consistency

In a widely-used approach to clustering, one selects $k$ of the input points as *centroids*, and then defines clusters by assigning each point in $S$ to its nearest centroid. The goal, intuitively, is to choose the centroids so that each point in $S$ is close to at least one of them. This overall approach arises both from combinatorial optimization perspectives, where it has roots in facility location problems [9], and in maximum-likelihood methods, where the centroids may represent centers of probability density functions [4, 6]. We show here that for a fairly general class of centroid-based clustering functions, including $k$-means and $k$-median, none of the functions in the class satisfies the Consistency property. This suggests an interesting tension between between Consistency and the centroid-based approach to clustering, and forms a contrast with the results for single-linkage and sum-of-pairs in previous sections.

Specifically, for any natural number $k \geq 2$, and any continuous, non-decreasing, and unbounded function $g : \mathbf{R}^+ \to \mathbf{R}^+$, we define the $(k, g)$-*centroid* clustering function as follows. First, we choose the set of $k$ "centroid" points $T \subseteq S$ for which the objective function $\Lambda_d^g(T) = \sum_{i \in S} g(d(i, T))$ is minimized. (Here $d(i, T) = \min_{j \in T} d(i, j)$.) Then we define a partition of $S$ into $k$ clusters by assigning each point to the element of $T$ closest to it. The $k$-median function [9] is obtained by setting $g$ to be the identity function, while the objective function underlying $k$-means clustering [4, 6] is obtained by setting $g(d) = d^2$.

**Theorem 4.1** *For every $k \geq 2$ and every function $g$ chosen as above, and for $n$ sufficiently large relative to $k$, the $(k, g)$-centroid clustering function does not satisfy the Consistency property.*

*Proof Sketch.* We describe the proof for $k = 2$ clusters; the case of $k > 2$ is similar. We consider a set of points $S$ that is divided into two subsets: a set $X$ consisting of $m$ points, and a set $Y$ consisting of $\gamma m$ points, for a small number $\gamma > 0$. The distance between points in $X$ is $r$, the distance between points in $Y$ is $\varepsilon < r$, and the distance from a point in $X$ to a point in $Y$ is $r + \delta$, for a small number $\delta > 0$. By choosing $\gamma$, $r$, $\varepsilon$, and $\delta$ appropriately, the optimal choice of $k = 2$ centroids will consist of one point from $X$ and one from $Y$, and the resulting partition $\Gamma$ will have clusters $X$ and $Y$. Now, suppose we divide $X$ into sets $X_0$ and $X_1$ of equal size, and reduce the distances between points in the same $X_i$ to be $r' < r$ (keeping all other distances the same). This can be done, for $r'$ small enough, so that the optimal choice of two centroids will now consist of one point from each $X_i$, yielding a different partition of $S$. As our second distance function is a $\Gamma$-transformation of the first, this violates Consistency. ■

## 5    Relaxing the Properties

In addition to looking for clustering functions that satisfy subsets of the basic properties, we can also study the effect of relaxing the properties themselves. Theorem 3.2 is a step in this direction, showing that the sum-of-pairs function satisfies Scale-Invariance and Consistency, together with a relaxation of the Richness property. As an another example, it is interesting to note that single-linkage with the distance-$r$ stopping condition satisfies a natural relaxation of Scale-Invariance: if $\alpha > 1$, then $f(\alpha \cdot d)$ is a refinement of $f(d)$.

We now consider some relaxations of Consistency. Let $f$ be a clustering function, and $d$ a distance function such that $f(d) = \Gamma$. If we reduce distances within clusters and expand distances between clusters, Consistency requires that $f$ output the same partition $\Gamma$. But one could imagine requiring something less: perhaps changing distances this way should be allowed to create additional sub-structure, leading to a new partition in which each cluster is a subset of one of the original clusters. Thus, we can define *Refinement-Consistency*, a relaxation of Consistency, to require that if $d'$ is an $f(d)$-transformation of $d$, then $f(d')$ should be a refinement of $f(d)$.

We can show that the natural analogue of Theorem 2.1 still holds: there is no clustering function that satisfies Scale-Invariance, Richness, and Refinement-Consistency. However, there is a crucial sense in which this result "just barely" holds, rendering it arguably less interesting to us here. Specifically, let $\Gamma_n^*$ denote the partition of $S = \{1, 2, \ldots, n\}$ in which each individual element forms its own cluster. Then there exist clustering functions $f$ that satisfy Scale-Invariance and Refinement-Consistency, and for which Range$(f)$ consists of all partitions except $\Gamma_n^*$. (One example is single-linkage with the distance-$(\alpha\delta)$ stopping condition, where $\delta = \min_{i,j} d(i,j)$ is the minimum inter-point distance, and $\alpha \geq 1$.) Such functions $f$, in addition to Scale-Invariance and Refinement-Consistency, thus satisfy a kind of *Near-Richness* property: one can obtain every partition as output except for a single, trivial partition. It is in this sense that our impossibility result for Refinement-Consistency, unlike Theorem 2.1, is quite "brittle."

To relax Consistency even further, we could say simply that if $d'$ is an $f(d)$-transformation of $d$, then one of $f(d)$ or $f(d')$ should be a refinement of the other. In other words, $f(d')$ may be *either* a refinement or a "coarsening" of $f(d)$. It is possible to construct clustering functions $f$ that satisfy this even weaker variant of Consistency, together with Scale-Invariance and Richness.

**Acknowledgements.** I thank Shai Ben-David, John Hopcroft, and Lillian Lee for valuable discussions on this topic. This research was supported in part by a David and Lucile Packard Foundation Fellowship, an ONR Young Investigator Award, an NSF Faculty Early Career Development Award, and NSF ITR Grant IIS-0081334.

# References

[1] M. Anderberg, *Cluster Analysis for Applications*, Academic Press, 1973.

[2] K. Arrow, *Social Choice and Individual Values*, Wiley, New York, 1951.

[3] M. Bern, D. Eppstein, "Approximation algorithms for geometric prolems," in *Approximation Algorithms for NP-Hard Problems*, (D. Hochbaum, Ed.), PWS Publishing, 1996.

[4] R. Duda, P. Hart, D. Stork, *Pattern Classification* (2nd edition), Wiley, 2001.

[5] P. Hansen, F. Roberts, "An impossibility result in axiomatic location theory," *Mathematics of Operations Research* 21(1996).

[6] A. Jain, R. Dubes, *Algorithms for Clustering Data*, Prentice-Hall, 1981.

[7] N. Jardine, R. Sibson, *Mathematical Taxonomy* Wiley, 1971.

[8] A. Kalai, C. Papadimitriou, S. Vempala, A. Vetta, personal communication, June 2002.

[9] P. Mirchandani, R. Francis, *Discrete Location Theory*, Wiley, 1990.

[10] M. Osborne A. Rubinstein, *A Course in Game Theory*, MIT Press, 1994.

[11] D. Pennock, E. Horvitz, C.L. Giles, "Social choice theory and recommender systems: Analysis of the axiomatic foundations of collaborative filtering," *Proc. 17th AAAI*, 2000.

[12] J. Puzicha, T. Hofmann, J. Buhmann "A Theory of Proximity Based Clustering: Structure Detection by Optimization," *Pattern Recognition*, 33(2000).
